# Worst-case Loss Bounds
# for Single Neurons

**David P. Helmbold**
Department of Computer Science
University of California, Santa Cruz
Santa Cruz, CA 95064
USA

**Jyrki Kivinen**
Department of Computer Science
P.O. Box 26 (Teollisuuskatu 23)
FIN-00014 University of Helsinki
Finland

**Manfred K. Warmuth**
Department of Computer Science
University of California, Santa Cruz
Santa Cruz, CA 95064
USA

## Abstract

We analyze and compare the well-known Gradient Descent algorithm and a new algorithm, called the Exponentiated Gradient algorithm, for training a single neuron with an arbitrary transfer function. Both algorithms are easily generalized to larger neural networks, and the generalization of Gradient Descent is the standard back-propagation algorithm. In this paper we prove worst-case loss bounds for both algorithms in the single neuron case. Since local minima make it difficult to prove worst-case bounds for gradient-based algorithms, we must use a loss function that prevents the formation of spurious local minima. We define such a matching loss function for any strictly increasing differentiable transfer function and prove worst-case loss bound for any such transfer function and its corresponding matching loss. For example, the matching loss for the identity function is the square loss and the matching loss for the logistic sigmoid is the entropic loss. The different structure of the bounds for the two algorithms indicates that the new algorithm out-performs Gradient Descent when the inputs contain a large number of irrelevant components.

# 1   INTRODUCTION

The basic element of a neural network, a *neuron*, takes in a number of real-valued input variables and produces a real-valued output. The input-output mapping of a neuron is defined by a *weight vector* $\mathbf{w} \in \mathbf{R}^N$, where $N$ is the number of input variables, and a *transfer function* $\phi$. When presented with input given by a vector $\mathbf{x} \in \mathbf{R}^N$, the neuron produces the output $\hat{y} = \phi(\mathbf{w} \cdot \mathbf{x})$. Thus, the weight vector regulates the influence of each input variable on the output, and the transfer function can produce nonlinearities into the input-output mapping. In particular, when the transfer function is the commonly used logistic function, $\phi(p) = 1/(1 + e^{-p})$, the outputs are bounded between 0 and 1. On the other hand, if the outputs should be unbounded, it is often convenient to use the identity function as the transfer function, in which case the neuron simply computes a linear mapping. In this paper we consider a large class of transfer functions that includes both the logistic function and the identity function, but not discontinuous (e.g. step) functions.

The goal of *learning* is to come up with a weight vector $\mathbf{w}$ that produces a desirable input-output mapping. This is achieved by considering a sequence $S = ((\mathbf{x}_1, y_1), \ldots, (\mathbf{x}_\ell, y_\ell))$ of *examples*, where for $t = 1, \ldots, \ell$ the value $y_t \in \mathbf{R}$ is the desired output for the input vector $\mathbf{x}_t$, possibly distorted by noise or other errors. We call $\mathbf{x}_t$ the $t$th *instance* and $y_t$ the $t$th *outcome*. In what is often called batch learning, all $\ell$ examples are given at once and are available during the whole training session. As noise and other problems often make it impossible to find a weight vector $\mathbf{w}$ that would satisfy $\phi(\mathbf{w} \cdot \mathbf{x}_t) = y_t$ for all $t$, one instead introduces a *loss function* $L$, such as the *square loss* given by $L(y, \hat{y}) = (y - \hat{y})^2/2$, and finds a weight vector $\mathbf{w}$ that minimizes the empirical loss (or training error)

$$\text{Loss}(\mathbf{w}, S) = \sum_{t=1}^{\ell} L(y_t, \phi(\mathbf{w} \cdot \mathbf{x}_t)) \ . \tag{1}$$

With the square loss and identity transfer function $\phi(p) = p$, this is the well-known linear regression problem. When $\phi$ is the logistic function and $L$ is the *entropic loss* given by $L(y, \hat{y}) = y \ln(y/\hat{y}) + (1 - y) \ln((1 - y)/(1 - \hat{y}))$, this can be seen as a special case of logistic regression. (With the entropic loss, we assume $0 \le y_t, \hat{y}_t \le 1$ for all $t$, and use the convention $0 \ln 0 = 0 \ln(0/0) = 0$.)

In this paper we use an *on-line prediction* (or life-long learning) approach to the learning problem. It is well known that on-line performance is closely related to batch learning performance (Littlestone, 1989; Kivinen and Warmuth, 1994). Instead of receiving all the examples at once, the training algorithm begins with some fixed start vector $\mathbf{w}_1$, and produces a sequence $\mathbf{w}_1, \ldots, \mathbf{w}_{\ell+1}$ of weight vectors. The new weight vector $\mathbf{w}_{t+1}$ is obtained by applying a simple *update rule* to the previous weight vector $\mathbf{w}_t$ and the single example $(\mathbf{x}_t, y_t)$. In the on-line prediction model, the algorithm uses its $t$th weight vector, or *hypothesis*, to make the *prediction* $\hat{y}_t = \phi(\mathbf{w}_t \cdot \mathbf{x}_t)$. The training algorithm is then charged a loss $L(y_t, \hat{y}_t)$ for this $t$th *trial*. The performance of a training algorithm $A$ that produces the weight vectors $\mathbf{w}_t$ on an example sequence $S$ is measured by its total (cumulative) loss

$$\text{Loss}(A, S) = \sum_{t=1}^{\ell} L(y_t, \phi(\mathbf{w}_t \cdot \mathbf{x}_t)) \ . \tag{2}$$

Our main results are bounds on the cumulative losses for two on-line prediction algorithms. One of these is the standard *Gradient Descent* (GD) algorithm. The other one, which we call $\text{EG}^\pm$, is also based on the gradient but uses it in a different

manner than GD. The bounds are derived in a worst-case setting: we make no assumptions about how the instances are distributed or the relationship between each instance $\mathbf{x}_t$ and its corresponding outcome $y_t$. Obviously, some assumptions are needed in order to obtain meaningful bounds. The approach we take is to compare the total losses, Loss(GD, $S$) and Loss(EG$^{\pm}$, $S$), to the least achievable empirical loss, $\inf_{\mathbf{w}}$ Loss($\mathbf{w}$, $S$). If the least achievable empirical loss is high, the dependence between the instances and outcomes in $S$ cannot be tracked by any neuron using the transfer function, so it is reasonable that the losses of the algorithms are also high. More interestingly, if some weight vector achieves a low empirical loss, we also require that the losses of the algorithms are low. Hence, although the algorithms always predict based on an initial segment of the example sequence, they must perform almost as well as the best fixed weight vector for the whole sequence.

The choice of loss function is crucial for the results that we prove. In particular, since we are using gradient-based algorithms, the empirical loss should not have spurious local minima. This can be achieved for any differentiable increasing transfer function $\phi$ by using the loss function $L_{\phi}$ defined by

$$L_{\phi}(y, \hat{y}) = \int_{\phi^{-1}(y)}^{\phi^{-1}(\hat{y})} (\phi(z) - y)\, dz \ . \tag{3}$$

For $y < \hat{y}$ the value $L_{\phi}(y, \hat{y})$ is the area in the $z \times \phi(z)$ plane below the function $\phi(z)$, above the line $\phi(z) = y$, and to the left of the line $z = \phi^{-1}(\hat{y})$. We call $L_{\phi}$ the *matching* loss function for transfer function $\phi$, and will show that for any example sequence $S$, if $L = L_{\phi}$ then the mapping from $\mathbf{w}$ to Loss($\mathbf{w}$, $S$) is convex. For example, if the transfer function is the logistic function, the matching loss function is the entropic loss, and if the transfer function is the identity function, the matching loss function is the square loss. Note that using the logistic activation function with the square loss can lead to a very large number of local minima (Auer et al., 1996). Even in the batch setting there are reasons to use the entropic loss with the logistic transfer function (see, for example, Solla et al., 1988).

How much our bounds on the losses of the two algorithms exceed the least empirical loss depends on the maximum slope of the transfer function we use. More importantly, they depend on various norms of the instances and the vector $\mathbf{w}$ for which the least empirical loss is achieved. As one might expect, neither of the algorithms is uniformly better than the other. Interestingly, the new EG$^{\pm}$ algorithm is better when most of the input variables are irrelevant, i.e., when some weight vector $\mathbf{w}$ with $w_i = 0$ for most indices $i$ has a low empirical loss. On the other hand, the GD algorithm is better when the weight vectors with low empirical loss have many nonzero components, but the instances contain many zero components.

The bounds we derive concern only single neurons, and one often combines a number of neurons into a multilayer feedforward neural network. In particular, applying the Gradient Descent algorithm in the multilayer setting gives the famous back propagation algorithm. Also the EG$^{\pm}$ algorithm, being gradient-based, can easily be generalized for multilayer feedforward networks. Although it seems unlikely that our loss bounds will generalize to multilayer networks, we believe that the intuition gained from the single neuron case will provide useful insight into the relative performance of the two algorithms in the multilayer case. Furthermore, the EG$^{\pm}$ algorithm is less sensitive to large numbers of irrelevant attributes. Thus it might be possible to avoid multilayer networks by introducing many new inputs, each of which is a non-linear function of the original inputs. Multilayer networks remain an interesting area for future study.

Our work follows the path opened by Littlestone (1988) with his work on learning

thresholded neurons with sparse weight vectors. More immediately, this paper is preceded by results on linear neurons using the identity transfer function (Cesa-Bianchi et al., 1996; Kivinen and Warmuth, 1994).

## 2   THE ALGORITHMS

This section describes how the Gradient Descent training algorithm and the new Exponentiated Gradient training algorithm update the neuron's weight vector.

For the remainder of this paper, we assume that the transfer function $\phi$ is increasing and differentiable, and $Z$ is a constant such that $\phi'(p) \leq Z$ holds for all $p \in \mathbf{R}$. For the loss function $L_\phi$ defined by (3) we have

$$\frac{\partial L_\phi(y, \phi(\mathbf{w} \cdot \mathbf{x}))}{\partial w_i} = (\phi(\mathbf{w} \cdot \mathbf{x}) - y)x_i \ . \tag{4}$$

Treating $L_\phi(y, \phi(\mathbf{w} \cdot \mathbf{x}))$ for fixed $\mathbf{x}$ and $y$ as a function of $\mathbf{w}$, we see that the Hessian $H$ of the function is given by $H_{ij} = \phi'(\mathbf{w} \cdot \mathbf{x})x_i x_j$. Then $\mathbf{v}^{\mathrm{T}} H \mathbf{v} = \phi'(\mathbf{w} \cdot \mathbf{x})(\mathbf{v} \cdot \mathbf{x})^2$, so $H$ is positive definite. Hence, for an arbitrary fixed $S$, the empirical loss $\mathrm{Loss}(\mathbf{w}, S)$ defined in (1) as a function of $\mathbf{w}$ is convex and thus has no spurious local minima.

We first describe the Gradient Descent (GD) algorithm, which for multilayer networks leads to the back-propagation algorithm. Recall that the algorithm's prediction at trial $t$ is $\hat{y}_t = \phi(\mathbf{w}_t \cdot \mathbf{x}_t)$, where $\mathbf{w}_t$ is the current weight vector and $\mathbf{x}_t$ is the input vector. By (4), performing gradient descent in weight space on the loss incurred in a single trial leads to the update rule

$$\mathbf{w}_{t+1} = \mathbf{w}_t - \eta(\hat{y}_t - y_t)\mathbf{x}_t \ .$$

The parameter $\eta$ is a positive *learning rate* that multiplies the gradient of the loss function with respect to the weight vector $\mathbf{w}_t$. In order to obtain worst-case loss bounds, we must carefully choose the learning rate $\eta$. Note that the weight vector $\mathbf{w}_t$ of GD always satisfies $\mathbf{w}_t = \mathbf{w}_1 + \sum_{i=1}^{t-1} a_i \mathbf{x}_i$ for some scalar coefficients $a_i$. Typically, one uses the zero initial vector $\mathbf{w}_1 = \mathbf{0}$.

A more recent training algorithm, called the Exponentiated Gradient (EG) algorithm (Kivinen and Warmuth, 1994), uses the same gradient in a different way. This algorithm makes multiplicative (rather than additive) changes to the weight vector, and the gradient appears in the exponent. The basic version of the EG algorithm also normalizes the weight vector, so the update is given by

$$w_{t+1,i} = w_{t,i} e^{-\eta(\hat{y}_t - y_t)x_{t,i}} \Big/ \sum_{j=1}^{N} w_{t,j} e^{-\eta(\hat{y}_t - y_t)x_{t,j}} \ .$$

The start vector is usually chosen to be uniform, $\mathbf{w}_1 = (1/N, \dots, 1/N)$. Notice that it is the logarithms of the weights produced by the EG training algorithm (rather than the weights themselves) that are essentially linear combinations of the past examples. As can be seen from the update, the EG algorithm maintains the constraints $w_{t,i} > 0$ and $\sum_i w_{t,i} = 1$. In general, of course, we do not expect that such constraints are useful. Hence, we introduce a modified algorithm $\mathrm{EG}^\pm$ by employing a linear transformation of the inputs. In addition to the learning rate $\eta$, the $\mathrm{EG}^\pm$ algorithm has a *scaling factor* $U > 0$ as a parameter. We define the behavior of $\mathrm{EG}^\pm$ on a sequence of examples $S = ((\mathbf{x}_1, y_1), \dots, (\mathbf{x}_\ell, y_\ell))$ in terms of the EG algorithm's behavior on a transformed example sequence $S' = ((\mathbf{x}'_1, y_1), \dots, (\mathbf{x}'_\ell, y_\ell))$

where $\mathbf{x}' = (U x_1, \ldots, U x_N, -U x_1, \ldots, -U x_N)$. The EG algorithm uses the uniform start vector $(1/(2N), \ldots, 1/(2N))$ and learning rate supplied by the $EG^{\pm}$ algorithm. At each time time $t$ the $N$-dimensional weight vector $\mathbf{w}$ of $EG^{\pm}$ is defined in terms of the $2N$-dimensional weight vector $\mathbf{w}'$ of EG as

$$w_{t,i} = U(w'_{t,i} - w'_{t,N+i}).$$

Thus $EG^{\pm}$ with scaling factor $U$ can learn any weight vector $\mathbf{w} \in \mathbf{R}^N$ with $\|\mathbf{w}\|_1 < U$ by having the embedded EG algorithm learn the appropriate $2N$-dimensional (nonnegative and normalized) weight vector $\mathbf{w}'$.

# 3   MAIN RESULTS

The loss bounds for the GD and $EG^{\pm}$ algorithms can be written in similar forms that emphasize how different algorithms work well for different problems. When $L = L_\phi$, we write $\mathrm{Loss}_\phi(\mathbf{w}, S)$ and $\mathrm{Loss}_\phi(A, S)$ for the empirical loss of a weight vector $\mathbf{w}$ and the total loss of an algorithm $A$, as defined in (1) and (2). We give the upper bounds in terms of various norms. For $\mathbf{x} \in \mathbf{R}^N$, the 2-norm $\|\mathbf{x}\|_2$ is the Euclidean length of the vector $\mathbf{x}$, the 1-norm $\|\mathbf{x}\|_1$ the sum of the absolute values of the components of $\mathbf{x}$, and the $\infty$-norm $\|\mathbf{x}\|_\infty$ the maximum absolute value of any component of $\mathbf{x}$. For the purposes of setting the learning rates, we assume that before training begins the algorithm gets an upper bound for the norms of instances. The GD algorithm gets a parameter $X_2$ and EG a parameter $X_\infty$ such that $\|\mathbf{x}_t\|_2 \le X_2$ and $\|\mathbf{x}_t\|_\infty \le X_\infty$ hold for all $t$. Finally, recall that $Z$ is an upper bound on $\phi'(p)$. We can take $Z = 1$ when $\phi$ is the identity function and $Z = 1/4$ when $\phi$ is the logistic function.

Our first upper bound is for GD. For any sequence of examples $S$ and any weight vector $\mathbf{u} \in \mathbf{R}^N$, when the learning rate is $\eta = 1/(2X_2^2 Z)$ we have

$$\mathrm{Loss}_\phi(GD, S) \le 2\mathrm{Loss}_\phi(\mathbf{u}, S) + 2(\|\mathbf{u}\|_2 X_2)^2 Z \ .$$

Our upper bounds on the $EG^{\pm}$ algorithm require that we restrict the one-norm of the *comparison class*: the set of weight vectors competed against. The comparison class contains all weight vectors $\mathbf{u}$ such that $\|\mathbf{u}\|_1$ is at most the scaling factor, $U$. For any scaling factor $U$, any sequence of examples $S$, and any weight vector $\mathbf{u} \in \mathbf{R}^N$ with $\|\mathbf{u}\|_1 \le U$, we have

$$\mathrm{Loss}_\phi(EG^{\pm}, S) \le \frac{4}{3}\mathrm{Loss}_\phi(\mathbf{u}, S) + \frac{16}{3}(U X_\infty)^2 Z \ln(2N)$$

when the learning rate is $\eta = 1/(4(U X_\infty)^2 Z)$.

Note that these bounds depend on both the unknown weight vector $\mathbf{u}$ and some norms of the input vectors. If the algorithms have some further prior information on the sequence $S$ they can make a more informed choice of $\eta$. This leads to bounds with a constant of 1 before the the $\mathrm{Loss}_\phi(\mathbf{u}, S)$ term at the cost of an additional square-root term (for details see the full paper, Helmbold et al., 1996).

It is important to realize that we bound the total loss of the algorithms over *any* adversarially chosen sequence of examples where the input vectors satisfy the norm bound. Although we state the bounds in terms of loss on the data, they imply that the algorithms must also perform well on new unseen examples, since the bounds still hold when an adversary adds these additional examples to the end of the sequence. A formal treatment of this appears in several places (Littlestone, 1989;

Kivinen and Warmuth, 1994). Furthermore, in contrast to standard convergence proofs (e.g. Luenberger, 1984), we bound the loss on the *entire* sequence of examples instead of studying the convergence behavior of the algorithm when it is arbitrarily close to the best weight vector.

Comparing these loss bounds we see that the bound for the $EG^\pm$ algorithm grows with the maximum component of the input vectors and the one-norm of the best weight vector from the comparison class. On the other hand, the loss bound for the GD algorithm grows with the two-norm (Euclidean length) of both vectors. Thus when the best weight vector is sparse, having few significant components, and the input vectors are dense, with several similarly-sized components, the bound for the $EG^\pm$ algorithm is better than the bound for the GD algorithm. More formally, consider the noise-free situation where $\text{Loss}_\phi(\mathbf{u}, S) = 0$ for some $\mathbf{u}$. Assume $\mathbf{x}_t \in \{-1, 1\}^N$ and $\mathbf{u} \in \{-1, 0, 1\}^N$ with only $k$ nonzero components in $\mathbf{u}$. We can then take $X_2 = \sqrt{N}$, $X_\infty = 1$, $\|\mathbf{u}\|_2 = \sqrt{k}$, and $U = k$. The loss bounds become $(16/3)k^2 Z \ln(2N)$ for $EG^\pm$ and $2kZN$ for GD, so for $N \gg k$ the $EG^\pm$ algorithm clearly wins this comparison. On the other hand, the GD algorithm has the advantage over the EG algorithm when each input vector is sparse and the best weight vector is dense, having its weight distributed evenly over its components. For example, if the inputs $\mathbf{x}_t$ are the rows of an $N \times N$ unit matrix and $\mathbf{u} \in \{-1, 1\}^N$, then $X_2 = X_\infty = 1$, $\|\mathbf{u}\|_2 = \sqrt{N}$, and $U = N$. Thus the upper bounds become $(16/3)N^2 Z \ln(2N)$ for $EG^\pm$ and $2NZ$ for GD, so here GD wins the comparison.

Of course, a comparison of the upper bounds is meaningless unless the bounds are known to be reasonably tight. Our experiments with artificial random data suggest that the upper bounds are not tight. However, the experimental evidence also indicates that $EG^\pm$ is much better than GD when the best weight vector is sparse. Thus the upper bounds do predict the relative behaviors of the algorithms.

The bounds we give in this paper are very similar to the bounds Kivinen and Warmuth (1994) obtained for the comparison class of linear functions and the square loss. They observed how the relative performances of the GD and $EG^\pm$ algorithms relate to the norms of the input vectors and the best weight vector in the linear case.

Our methods are direct generalizations of those applied for the linear case (Kivinen and Warmuth, 1994). The key notion here is a *distance function d* for measuring the distance $d(\mathbf{u}, \mathbf{w})$ between two weight vectors $\mathbf{u}$ and $\mathbf{w}$. Our main distance measures are the Squared Euclidean distance $\frac{1}{2}\|\mathbf{u} - \mathbf{w}\|_2^2$ and the Relative Entropy distance (or Kullback-Leibler divergence) $\sum_{i=1}^N u_i \ln(u_i/w_i)$. The analysis exploits an invariant over $t$ and $\mathbf{u}$ of the form

$$aL_\phi(y_t, \mathbf{w}_t \cdot \mathbf{x}_t) - bL_\phi(y_t, \mathbf{u} \cdot \mathbf{x}_t) \leq d(\mathbf{u}, \mathbf{w}_t) - d(\mathbf{u}, \mathbf{w}_{t+1}) \ ,$$

where $a$ and $b$ are suitably chosen constants. This invariant implies that at each trial, if the loss of the algorithm is much larger than that of an arbitrary vector $\mathbf{u}$, then the algorithm updates its weight vector so that it gets closer to $\mathbf{u}$. By summing the invariant over all trials we can bound the total loss of the algorithms in terms of $\text{Loss}_\phi(\mathbf{u}, S)$ and $d(\mathbf{u}, \mathbf{w}_1)$. Full details will be contained in a technical report (Helmbold et al., 1996).

# 4   OPEN PROBLEMS

Although the presence of local minima in multilayer networks makes it difficult to obtain worst case bounds for gradient-based algorithms, it may be possible to

analyze slightly more complicated settings than just a single neuron. One likely candidate is to generalize the analysis to logistic regression with more than two classes. In this case each class would be represented by one neuron.

As noted above, the matching loss for the logistic transfer function is the entropic loss, so this pair does not create local minima. No bounded transfer function matches the square loss in this sense (Auer et al., 1996), and thus it seems impossible to get the same kind of strong loss bounds for a bounded transfer function and the square loss as we have for any (increasing and differentiable) transfer function and its matching loss function.

As the bounds for $EG^{\pm}$ depend only logarithmically on the input dimension, the following approach may be feasible. Instead of using a multilayer net, use a single (linear or sigmoided) neuron on top of a large set of basis functions. The logarithmic growth of the loss bounds in the number of such basis functions mean that large numbers of basis functions can be tried.

Note that the bounds of this paper are only worst-case bounds and our experiments on artificial data indicate that the bounds may not be tight when the input values and best weights are large. However, we feel that the bounds do indicate the relative merits of the algorithms in different situations. Further research needs to be done to tighten the bounds. Nevertheless, this paper gives the first worst-case upper bounds for neurons with nonlinear transfer functions.

## References

P. Auer, M. Herbster, and M. K. Warmuth (1996). Exponentially many local minima for single neurons. In *Advances in Neural Information Processing Systems 8*.

N. Cesa-Bianchi, P. Long, and M. K. Warmuth (1996). Worst-case quadratic loss bounds for on-line prediction of linear functions by gradient descent. *IEEE Transactions on Neural Networks*. To appear. An extended abstract appeared in *COLT '93*, pp. 429–438.

D. P. Helmbold, J. Kivinen, and M. K. Warmuth (1996). Worst-case loss bounds for single neurons. Technical Report UCSC-CRL-96-2, Univ. of Calif. Computer Research Lab, Santa Cruz, CA, 1996. In preparation.

J. Kivinen and M. K. Warmuth (1994). Exponentiated gradient versus gradient descent for linear predictors. Technical Report UCSC-CRL-94-16, Univ. of Calif. Computer Research Lab, Santa Cruz, CA, 1994. An extended abstract appeared in *STOC '95*, pp. 209-218.

N. Littlestone (1988). Learning when irrelevant attributes abound: A new linear-threshold algorithm. *Machine Learning*, 2:285–318.

N. Littlestone (1989). From on-line to batch learning. In *Proc. 2nd Annual Workshop on Computational Learning Theory*, pages 269–284. Morgan Kaufmann, San Mateo, CA.

D. G. Luenberger (1984). *Linear and Nonlinear Programming*. Addison-Wesley, Reading, MA.

S. A. Solla, E. Levin, and M. Fleisher (1988). Accelerated learning in layered neural networks. *Complex Systems*, 2:625–639.
